# Adaptive Multi-Task Lasso: with Application to eQTL Detection

**Seunghak Lee, Jun Zhu and Eric P. Xing**
School of Computer Science, Carnegie Mellon University
{seunghak,junzhu,epxing}@cs.cmu.edu

## Abstract

To understand the relationship between genomic variations among population and complex diseases, it is essential to detect eQTLs which are associated with phenotypic effects. However, detecting eQTLs remains a challenge due to complex underlying mechanisms and the very large number of genetic loci involved compared to the number of samples. Thus, to address the problem, it is desirable to take advantage of the structure of the data and prior information about genomic locations such as conservation scores and transcription factor binding sites.

In this paper, we propose a novel regularized regression approach for detecting eQTLs which takes into account related traits simultaneously while incorporating many regulatory features. We first present a Bayesian network for a multi-task learning problem that includes priors on SNPs, making it possible to estimate the significance of each covariate adaptively. Then we find the maximum a posteriori (MAP) estimation of regression coefficients and estimate weights of covariates jointly. This optimization procedure is efficient since it can be achieved by using a projected gradient descent and a coordinate descent procedure iteratively. Experimental results on simulated and real yeast datasets confirm that our model outperforms previous methods for finding eQTLs.

## 1 Introduction

One of the fundamental problems in computational biology is to understand associations between genomic variations and phenotypic effects. The most common genetic variations are single nucleotide polymorphisms (SNPs), and many association studies have been conducted to find SNPs that cause phenotypic variations such as diseases or gene-expression traits [1]. However, association mapping of causal QTLs or eQTLs remains challenging as the variation of complex traits is a result of contributions of many genomic variations. In this paper, we focus on two important problems to detect eQTLs. First, we need to find methods to take advantage of the structure of data for finding association SNPs from high dimensional eQTL datasets when $p \gg N$, where $p$ is the number of SNPs and $N$ is the sample size. Second, we need techniques to take advantage of prior biological knowledge to improve the performance of detecting eQTLs.

To address the first problem, Lasso is a widely used technique for high-dimensional association mapping problems, which can yield a sparse and easily interpretable solution via an $\ell_1$ regularization [2]. However, despite the success of Lasso, it is limited to considering each trait separately. If we have multiple related traits it would be beneficial to estimate eQTLs jointly since we can share information among related traits. For the second problem, Fig. 1 shows some prior knowledge on SNPs in a genome including transcription factor binding sites (TFBS), 5' UTR and exon, which play important roles for the regulation of genes. For example, TFBS controls the transcription of DNA sequences to mRNAs. Intuitively, if SNPs are located on these regions, they are more likely to be true eQTLs compared to those on regions without such annotations since they are related to genes or gene regulations. Thus, it would be desirable to penalize regression coefficients less corresponding

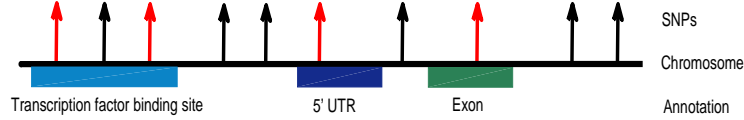

Figure 1: Examples of prior knowledge on SNPs including transcription factor binding sites, 5' UTR and exon. Arrows represent SNPs and we indicate three genomic annotations on the chromosome. Here association SNPs are denoted by red arrows (best viewed in color), showing that SNPs on regions with regulatory features are more likely to be associated with traits.

to SNPs having significant annotations such as TFBS in a regularized regression model. Again, the widely used Lasso is limited to treating all SNPs equally.

This paper presents a novel regularized regression approach, called *adaptive multi-task Lasso*, to effectively incorporate both the relatedness among multiple gene-expression traits and useful prior knowledge for challenging eQTL detection. Although some methods have been developed for either adaptive or multi-task learning, to the best of our knowledge, adaptive multi-task Lasso is the first method that can consider prior information on SNPs and multi-task learning simultaneously in one single framework. For example, Lirnet uses prior knowledge on SNPs such as conservation scores, non-synonymous coding and UTR regions for a better search of association mappings [3]. However, Lirnet considers the average effects of SNPs on gene modules by assuming that association SNPs are shared in a module. This approach is different from multi-task learning where association SNPs are found for each trait while considering group effects over multiple traits. To find genetic markers that affect correlated traits jointly, the graph-guided fused Lasso [4] was proposed to consider networks over multiple traits within an association analysis. However, graph-guided fused Lasso does not incorporate prior knowledge of genomic locations.

Unlike other methods, we define the adaptive multi-task Lasso as finding a MAP estimate of a Bayesian network, which provides an elegant Bayesian interpretation of our approach; the resultant optimization problem is efficiently solved with an alternating minimization procedure. Finally, we present empirical results on both simulated and real yeast eQTL datasets, which demonstrates the advantages of adaptive multi-task Lasso over many other competitors.

## 2 Problem Definition: Adaptive Multi-task Lasso

Let $X_{ij} \in \{0, 1, 2\}$ denote the number of minor alleles at the $j$-th SNP of $i$-th individual for $i = 1, \ldots, N$ and $j = 1, \ldots, p$. We have $K$ related gene traits and $Y_i^k$ represents the gene expression level of $k$-th gene of $i$-th individual for $k = 1, \ldots, K$. In our setting, we assume that the $K$ traits are related to each other and we explore the relatedness in a multi-task learning framework. To achieve the relatedness among tasks via grouping effects [5], we can use any clustering algorithms such as spectral clustering or hierarchical clustering. In association mapping problems, these clusters can be viewed as clusters of genes which consist of regulatory networks or pathways [4]. We treat the problem of detecting eQTLs as a linear regression problem. The general setting includes one design matrix $X$ and multiple tasks (genes) for $k = 1, \ldots, K$,

$$Y^k = X\beta^k + \epsilon \tag{1}$$

where $\epsilon$ is a standard Gaussian noise. We further assume that $X_{ij}$'s are standardized such that $\sum_i X_{ij}/N = 0$ and $\sum_i X_{ij}^2/N = 1$, and consider a model without an intercept.

Now, the open question is how we can devise an appropriate objective function over $\beta$ that could effectively consider the desirable group effects over multiple traits and incorporate useful prior knowledge, as we have stated. To explain the motivation of our work and provide a useful baseline that grounds the proposed approach, we first briefly review the standard Lasso and multi-task Lasso.

### 2.1 Lasso and Multi-task Lasso

Lasso [2] is a technique for estimating the regression coefficients $\beta$ and has been widely used for association mapping problems. Mathematically, it solves the $\ell_1$-regularized least square problem,

$$\hat{\beta} = \underset{\beta}{\text{argmin}} \frac{1}{2}\|Y - X\beta\|_2^2 + \lambda \sum_{j=1}^{p} \delta_j |\beta_j| \tag{2}$$

where $\lambda$ determines the degree of regularization of nonzero $\beta_j$. The scaling parameters $\delta_j \in [0, 1]$ are usually fixed (e.g., unit ones) or set by cross-validation, which can be very difficult when $p$ is large. Due to the singularity at the origin, the $\ell_1$ regularization (Lasso penalty) can yield a stable and sparse solution, which is desirable for association mapping problems because in most cases we have $p \gg N$ and there exists only a small number of eQTLs. It is worth mentioning that Lasso estimates are posterior mode estimates under a multivariate independent Laplace prior for $\beta$ [2].

As we can see from problem (2), the standard Lasso does not distinguish the inputs and regression coefficients from different tasks. In order to capture some desirable properties (e.g., shared structures or sparse patterns) among multiple related tasks, the multi-task Lasso was proposed [5], which solves the problem,

$$\min_{\beta} \frac{1}{2} \sum_{k=1}^{K} \|Y^k - X\beta^k\|_2^2 + \lambda \sum_{j=1}^{p} \delta_j \|\beta_j\|_2 \tag{3}$$

where $\|\beta_j\|_2 = \sqrt{\sum_k (\beta_j^k)^2}$ is the $\ell_2$-norm. This model encourages group-wise sparsity across related tasks via the $\ell_1/\ell_2$ regularization. Again, the solution of Eq. (3) can be interpreted as a MAP estimate under appropriate priors with *fixed* scaling parameters.

Multi-task Lasso has been applied (with some extensions) to perform association analysis [4]. However, as we have stated, the limitation of current approaches is that they do not incorporate the useful prior knowledge. The proposed adaptive multi-task Lasso, as to be presented, is an extension of the multi-task Lasso to perform joint *group-wise* and *within-group* feature selection and incorporate the useful prior knowledge for effective association analysis.

## 2.2 Adaptive Multi-task Lasso

Now, we formally introduce the adaptive multi-task Lasso. For clarity, we first define the *sparse multi-task Lasso* with fixed scaling parameters, which will be a sub-problem of the adaptive multi-task Lasso, as we shall see. Specifically, sparse multi-task Lasso solves the problem,

$$\min_{\beta} \frac{1}{2} \sum_{k=1}^{K} \|Y^k - X\beta^k\|_2^2 + \lambda_1 \sum_{j=1}^{p} \theta_j \sum_{k=1}^{K} |\beta_j^k| + \lambda_2 \sum_{j=1}^{p} \rho_j \|\beta_j\|_2 \tag{4}$$

where $\theta$ and $\rho$ are the scaling parameters for the $\ell_1$ and $\ell_1/\ell_2$-norm, respectively. The regularization parameters $\lambda_1$ and $\lambda_2$ can be determined by cross or holdout validation. Obviously, this model subsumes the standard Lasso and multi-task Lasso, and it has three advantages over previous models. First, unlike the multi-task Lasso, which contains the $\ell_l/\ell_2$-norm only to achieve group-wise sparsity, the $\ell_1$-norm in Eq. (4) can achieve sparsity among SNPs *within* a group. This property is useful when $K$ tasks are not perfectly related and we need additional sparsity in each block of $\|\beta_j\|_2$. In section 4, we demonstrate the usefulness of the blended regularization. The hierarchical penalization [6] can achieve a *smooth* shrinkage effect for variables within a group, but it cannot achieve within-group sparsity. Second, unlike Lasso we induce group sparsity across multiple related traits. Finally, as to be extended, unlike Lasso and multi-task Lasso which treat $\beta_j$ equally or with a fixed scaling parameter, we can adaptively penalize each $\beta_j$ according to prior knowledge on covariates in such a way that SNPs having desirable features are less penalized (see Fig. 1 for details of prior knowledge on SNPs).

To incorporate the prior knowledge as we have stated, we propose to automatically learn the scaling parameters $(\theta, \rho)$ from data. To that end, we define $\theta$ and $\rho$ as mixtures of features on $j$-th SNP, i.e.

$$\theta_j = \sum_t \omega_t f_t^j \quad \text{and} \quad \rho_j = \sum_t \nu_t f_t^j, \tag{5}$$

where $f_t^j$ is $t$-th feature for $j$-th SNP. For example $f_t^j$ can be a conservation score of $j$-th SNP or one if the SNP is located on TFBS, zero otherwise. To avoid scaling issues, we assume each feature is standardized, i.e., $\sum_j f_t^j = 1$, $\forall t$. Since we are interested in the relative contributions from different features, we further add the constraints that $\sum_t \omega_t = 1$ and $\sum_t \nu_t = 1$. These constraints can be interpreted as a regularization on the feature weights $\omega \geq 0$ and $\nu \geq 0$.

Although using the definitions (5) in problem (4) and jointly estimating $\beta$ and feature weights $(\omega, \nu)$ can give a solution of adaptive multi-task learning, the resultant method would be lack of an elegant Bayesian interpretation, which is a desirable property that can make the framework more

flexible and easily extensible. Recall that the Lasso estimates can be interpreted as MAP estimates under Laplace priors. Similarly, to achieve a framework that enjoys an elegant Bayesian interpretation, we define a Bayesian network and treat the adaptive multi-task learning problem as finding its MAP estimate. Specifically, we build a Bayesian network as shown in Fig. 2 in order to compute the MAP estimate of $\beta$ under adaptive scaling parameters, $\{\theta, \rho\}$. We define the conditional probability of $\beta$ given scaling parameters as,

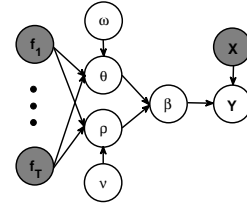

Figure 2: Graphical model representation of adaptive multi-task Lasso.

$$P(\beta|\theta,\rho) = \frac{1}{Z(\theta,\rho)} \prod_{j=1}^{p} \prod_{k=1}^{K} \exp\left(-\theta_j |\beta_j^k|\right) \times \prod_{j=1}^{p} \exp\left(-\rho_j \|\beta_j\|_2\right)$$

where $Z(\theta, \rho)$ is a normalization factor, and $P(Y|X, \beta) \sim N(X\beta, \Sigma)$, where $\Sigma$ is the identity matrix. Although in principle we can treat $\theta$ and $\rho$ as random variables and define a fully Bayesian approach, for simplicity, we define $\theta$ and $\rho$ as deterministic functions of $\omega$ and $\nu$ as in Eq. (5). Extension to a fully Bayesian approach is our future work.

Now we define the *adaptive multi-task Lasso* as finding the MAP estimation of $\beta$ and simultaneously estimating the feature weights $(\omega, \nu)$, which is equivalent to solving the optimization problem,

$$\min_{\beta,\omega,\nu} \frac{1}{2} \sum_{k=1}^{K} \|Y^k - X\beta^k\|_2^2 + \lambda_1 \sum_{j=1}^{p} \theta_j \sum_{k=1}^{K} |\beta_j^k| + \lambda_2 \sum_{j=1}^{p} \rho_j \|\beta_j\|_2 + \log Z(\theta, \rho), \qquad (6)$$

where $\omega$ and $\nu$ are related to $\theta$ and $\rho$ through Eq. (5) and subject to the constraints as defined above.

**Remark 1** *Although we can interpret problem (4) as a MAP estimate of $\beta$ under appropriate priors when scaling parameters $(\theta, \rho)$ are fixed, it does not enjoy an elegant Bayesian interpretation if we perform joint estimation of $\beta$ and the scaling parameters $(\omega, \nu)$ because it ignores normalization factors of the appropriate priors. Lee et al. [3] used this approach where a regularized regression model is optimized over scaling parameters and $\beta$ jointly. Therefore, their method does not have an elegant Bayesian interpretation. Moreover, as we have stated, Lee et al. [3] did not consider grouping effects over multiple traits.*

**Remark 2** *Our method also differs from the adaptive Lasso [7], transfer learning with meta-priors [8] and the Bayesian Lasso [9]. First, although both adaptive Lasso and our method use adaptive parameters for penalizing regression coefficients, we learn adaptive parameters from prior knowledge on covariates in a multi-task setting while adaptive Lasso uses ordinary least square solutions for adaptive parameters in a single task setting. Second, the method of transfer learning with meta-priors [8] is similar to our method in a sense that both use prior knowledge with multiple related tasks. However, we couple related tasks via $\ell_1/\ell_2$ penalty while they couple tasks via transferring hyper-parameters among them. Thus we have group sparsity across tasks as well as sparsity in each group but they cannot induce group sparsity across different tasks. Finally, the Bayesian Lasso [9] does not have the grouping effects in multiple traits and the priors used usually do not consider domain knowledge.*

## 3  Optimization: an Alternating Minimization Approach

Now, we solve the adaptive multi-task Lasso problem (6). First, since the normalization factor $Z$ is hard to compute, we use its upper bound, as given by,

$$Z \leq \prod_{j=1}^{p} \int_{\mathbb{R}^K} \exp\left(-\|\rho_j\|_2\right) d\rho \prod_j \left(\frac{2}{\theta_j}\right)^K = \prod_{j=1}^{p} \frac{\pi^{\frac{K-1}{2}} \Gamma(\frac{K+1}{2}) 2^K}{(\rho_j K)^K} \prod_j \left(\frac{2}{\theta_j}\right)^K. \qquad (7)$$

This integral result is due to normalization constant of $K$ dimensional multivariate Laplace distribution [10, 11]. Using this upper bound, the learning problem is to minimize an upper bound of the objective function in problem (6), which will be denoted by $\mathcal{L}(\beta, \omega, \nu)$ henceforth. Although $\mathcal{L}$ is not joint convex over $\beta$, $\omega$ and $\nu$, it is convex over $\beta$ given $\{\omega, \nu\}$ and convex over $\{\omega, \nu\}$ given $\beta$.

We use an alternating optimization procedure which (1) minimizes the upper bound $\mathcal{L}$ of problem (6) over $\{\omega, \nu\}$ by fixing $\beta$; and (2) minimizes $\mathcal{L}$ over $\beta$ by fixing $\{\omega, \nu\}$ iteratively until convergence [12]. Both sub-problems are convex and can be solved efficiently via a projected gradient descent method and a coordinate descent method, respectively.

For the first step of optimizing $\mathcal{L}$ over $\omega$ and $\nu$, the sub-problem is to solve

$$\min_{\omega \in \mathcal{P}_\omega, \nu \in \mathcal{P}_\nu} \sum_j \sum_k \left( -\log \theta_j + \theta_j |\beta_j^k| \right) + \sum_j \left( -K \log \rho_j + \rho_j \|\beta_j\|_2 \right),$$

where $\mathcal{P}_\omega \triangleq \{\omega : \sum_t \omega_t = 1, \ \omega_t \geq 0, \forall t\}$ is a simplex over $\omega$, likewise for $\mathcal{P}_\nu$. $\theta$ and $\rho$ are functions of $\omega$ and $\nu$ as defined in Eq. (5). This constrained problem is convex and can be solved by using a gradient descent algorithm combined with a projection onto a simplex sub-space, which can be efficiently done [13]. Since $\omega$ and $\nu$ are not coupled, we can learn each of them separately.

For the second sub-problem that optimizes $\mathcal{L}$ over $\beta$ given fixed feature weights $(\omega, \nu)$, it is exactly the optimization problem (4). We can solve it using a coordinate descent procedure, which has been used to optimize the sparse group Lasso [14]. Our problem is different from the sparse group Lasso in the sense that the sparse group Lasso includes group penalty over multiple covariates for a single trait, while adaptive multi-task Lasso considers group effects over multiple traits. Here we solve problem (4) using a modified version of the algorithm proposed for the sparse group Lasso.

As summarized in Algorithm 1, the general optimization procedure is as follows: for each $j$, we check the group sparsity condition that $\beta_j = 0$. If it is true, no update is needed for $\beta_j$. Otherwise, we check whether $\beta_j^k = 0$ for each $k$. If it is true that $\beta_j^k = 0$, no update is needed for $\beta_j^k$; otherwise, we optimize problem (4) over $\beta_j^k$ with all other coefficients fixed. This one-dimensional optimization problem can be efficiently solved by using a standard optimization method. This procedure is continued until a convergence condition is met.

More specifically, we first obtain the optimal conditions for problem (4) by computing the subgradient of its objective function with respect to $\beta_j^k$ and set it to zero:

$$-X_j^T (Y^k - X\beta^k) + \lambda_2 \rho_j g_j^k + \lambda_1 \theta_j h_j^k = 0, \tag{8}$$

where $g$ and $h$ are sub-gradients of the $\ell_1/\ell_2$-norm and the $\ell_1$-norm, respectively. Note that $g_j^k = \frac{\beta_j^k}{\|\beta_j\|_2}$ if $\beta_j \neq 0$, otherwise $\|g_j\|_2 \leq 1$; and $h_j^k = sign(\beta_j^k)$ if $\beta_j^k \neq 0$, otherwise $h_j^k \in [-1, 1]$.

Then, we check the group sparsity that $\beta_j = 0$. To do that, we set $\beta_j = 0$ in Eq. (8), and we have,

$$X_j^T Y^k - X_j^T \sum_{r \neq j} X_r \beta_r^k = \lambda_2 \rho_j g_j^k + \lambda_1 \theta_j h_j^k, \text{ and } \|g_j\|_2^2 = \frac{1}{\lambda_2^2 \rho_j^2} \sum_{k=1}^K (X_j^T Y^k - X_j^T \sum_{r \neq j} X_r \beta_r^k - \lambda_1 \theta_j h_j^k)^2.$$

According to subgradient conditions, we need to have a $g_j$ that satisfies the *less than* inequality $\|g_j\|_2^2 < 1$; otherwise, $\beta_j$ will be non-zero. Since $g_j$ is a function of $h_j$, it suffices to check whether the minimal square $\ell_2$-norm of $g_j$ is less than 1. Therefore, we solve the minimization problem of $\|g_j\|_2^2$ w.r.t $h_j$, which gives the optimal $h_j$ as,

$$h_j^k = \begin{cases} \frac{c_j^k}{\lambda_1 \theta_j} & \text{if } |\frac{c_j^k}{\lambda_1 \theta_j}| \leq 1 \\ sign(\frac{c_j^k}{\lambda_1 \theta_j}) & \text{otherwise} \end{cases} \tag{9}$$

where $c_j^k = X_j^T Y^k - X_j^T \sum_{r \neq j} X_r \beta_r^k$. If the minimal $\|g_j\|_2^2$ is less than 1, then $\beta_j$ is zero and no update is needed; otherwise, we continue to the next step of checking whether $\beta_j^k = 0$, $\forall k$, as follows.

Again, we start by assuming $\beta_j^k$ is zero. By setting $\beta_j^k = 0$ in Eq. (8), we have,

$$X_j^T Y^k - X_j^T \sum_{r \neq j} X_r \beta_r^k = \lambda_1 \theta_j h_j^k, \text{ and } h_j^k = \frac{1}{\lambda_1 \theta_j} (X_j^T Y^k - X_j^T \sum_{r \neq j} X_r \beta_r^k).$$

According to the definition of the subgradient $h_j^k$, it needs to satisfy the condition that $|h_j^k| < 1$; otherwise, $\beta_j^k$ will be non-zero. This checking step can be easily done. After the check, if we have $\beta_j^k \neq 0$, the problem (4) becomes an one-dimensional optimization problem with respect to $\beta_j^k$, and the solution can be obtained using existing optimization algorithms (e.g. optimize function in the R). We used majorize-minimize algorithm with gradient descent [15].

With the above two steps, we iteratively optimize $(\omega, \nu)$ by fixing $\beta$ and optimize $\beta$ by fixing feature weights until convergence. Note that the parameters $\lambda_1$ and $\lambda_2$ in Eq. (4), which determine sparsity levels, are determined by cross or hold-out validation.

---

**Input** : $X \in \mathbb{R}^{N \times p}; Y \in \mathbb{R}^{N \times K}; \theta \in \mathbb{R}^p; \rho \in \mathbb{R}^p;$ and $\beta^{\text{init}} \in \mathbb{R}^{p \times K}$
**Output**: $\beta \in \mathbb{R}^{p \times K}$

$\beta \leftarrow \beta^{init}$;

Iterate this procedure until convergence;

**for** $j \leftarrow 1$ **to** $p$ **do**

    $m \leftarrow \frac{1}{\lambda_2^2 \rho_j^2} \sum_{k=1}^K (c_j^k - \lambda_1 \theta_j h_j^k)^2$ where $c_j^k$ and $h_j^k$ are computed as in Eq. (9);

    **if** $m < 1$ **then** $\beta_j^k = 0$, for all $k = 1, \ldots K$;

    **else for** $k \leftarrow 1$ **to** $K$ **do**

        $q \leftarrow \frac{1}{\lambda_1 \theta_j} |X_j^T (Y^k - X\beta^k) + X_j^T X_j \beta_j^k|$;

        **if** $q < 1$ **then** $\beta_j^k = 0$;

        **else** Solve the following one-dimensional optimization problem:

        $\beta_j^k \leftarrow \underset{\beta_j^k}{\text{argmin}} \frac{1}{2}\|Y^k - X\beta^k\|_2^2 + \lambda_1 \theta_j |\beta_j^k| + \lambda_2 \rho_j \|\beta_j\|_2$;

    **end**

**end**

---

**Algorithm 1:** Optimization algorithm for Equation (4) with fixed scaling parameters.

## 4 Simulation Study

To confirm the behavior of our model, we run the adaptive multi-task Lasso and other methods on our simulated dataset (p=100, K=10). We first randomly select 100 SNPs from 114 yeast genotypes from the yeast eQTL dataset [16]. Following the simulation study in Kim et al. [4], we assume that some SNPs affect biological networks including multiple traits, and true causal SNPs are selected by the following procedure. Three sets of randomly selected four SNPs are associated with three trait clusters $(1-3)$, $(4-6)$, $(7-10)$, respectively. One SNP is associated with two clusters $(1-3)$ and $(4-6)$, and one causal SNP is for all traits $(1-10)$. For all association SNPs we set identical association strength from 0.3 to 1. Traits are generated by $Y^k = X\beta^k + \epsilon$, for all $k = 1, \ldots, 10$ where $\epsilon$ follows the standard normal distribution. We make 10 features $(f_1 - f_{10})$, of which six are continuous and four are discrete. For the first three continuous features $(f_1 - f_3)$, the feature value is drawn from $s(N(2,1))$ if a SNP is associated with any traits; otherwise from $s(N(1,1))$, where $s(x) = \frac{1}{1+\exp(x)}$ is the sigmoid function. For the other three continuous features $(f_4 - f_6)$, the value is drawn from $s(N(2,0.5))$ if a SNP is associated with any traits; otherwise from $s(N(1,0.5))$. Finally, for the discrete features $(f_7 - f_{10})$, the value is set to $s(2)$ with probability 0.8 if a SNP is associated with any traits; otherwise to $s(1)$. We standardize all the features.

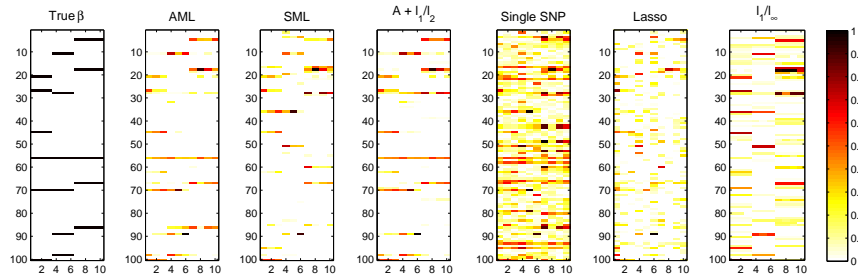

Figure 3: Results of the $\beta$ matrix estimated by different methods. For visualization, we present normalized absolute values of regression coefficients and darker colors imply stronger association with traits. For each matrix, X-axis represents traits (1-10) and Y-axis represents SNPs (1-100). True $\beta$ is shown in the left.

Fig. 3 shows the estimated $\beta$ matrix by various methods including AML (adaptive multi-task Lasso), SML (sparse multi-task Lasso which is AML without adaptive weights), A+$\ell_1/\ell_2$ (AML without Lasso penalty), Single SNP [17], Lasso and $\ell_1/\ell_\infty$ (multi-task learning with $\ell_1/\ell_\infty$ norm). In this figure, X-axis represents traits (1-10) and Y-axis represents SNPs (1-100). Note that regression parameters (e.g. $\lambda_1$ and $\lambda_2$ for AML) were determined by holdout validation, and we set association strength to 0.3. We also used hierarchical clustering with cutoff criterion 0.8 prior to run AML, SML, A+$\ell_1/\ell_2$ and $\ell_1/\ell_\infty$, and Single SNP and Lasso were analyzed for each trait separately.

We investigate the effect of Lasso penalty in our model by comparing the results of AML and A+$\ell_1/\ell_2$. While AML is slightly more efficient than A+$\ell_1/\ell_2$ in finding association SNPs, both

work very well for this task. It is not surprising since hierarchical clustering reproduced true trait clusters and true $\beta$ could be detected without considering single SNP level sparsity in each group. To further validate the effectiveness of Lasso penalty, we run AML and A+$\ell_1/\ell_2$ without a priori clustering step. Interestingly, AML could pick correct SNP-traits associations due to Lasso penalty, however, A+$\ell_1/\ell_2$ failed to do so (see Fig. 5c,d for the comparison of performance). While Lasso penalty did not show significant contribution for this task when we generated a priori clusters, it is good to include it when the quality of a clustering is not guaranteed. Comparing the results of AML and SML in Fig. 3, we could observe that adaptive weights improve the performance significantly. Adaptive weights help not only reduce false positives but also increase true positives.

Fig. 4 shows the learned feature weights of $\omega$ ($\nu$ is almost identical to $\omega$ and not shown here). The results are based on 100 simulations for each association strength 0.3, 0.5, 0.8 and 1, and half of error bar represents one standard deviation from the mean. We could observe that discrete features $f_7 - f_{10}$ have highest weights while lowest weights are assigned to $f_1 - f_3$. These weights are reasonable because $f_1 - f_3$ are drawn from Gaussian with large standard deviation (STD: 1) compared to that of features $f_4 - f_6$ (STD: 0.5). Also, discrete features are the most important since they discriminate true association SNPs with a high probability 0.8.

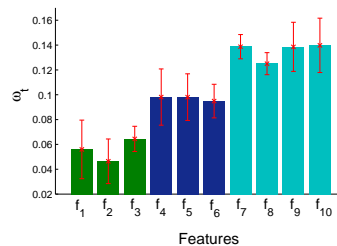

Figure 4: Learned feature weights of $\omega$.

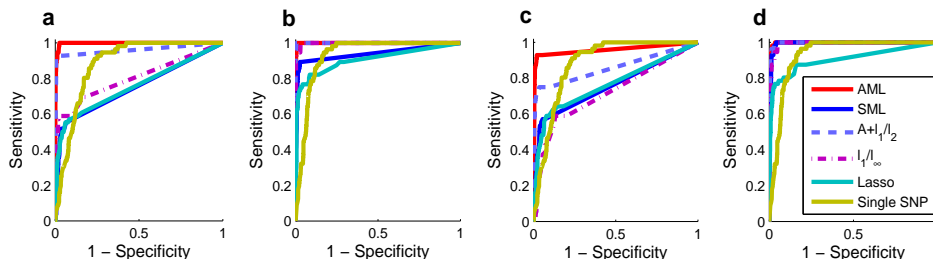

Figure 5: ROC curves of various methods as association strength varies (a) 0.3, (b) 0.5 on clustered data, (c) 0.3, and (d) 0.5 on input dataset. (a,b) Results on clustered data, where correct groups of gene traits are found using hierarchical clustering (cutoff = 0.8). (c,d) Results on input dataset without using clustering algorithm.

We compare the sensitivity and specificity of our model with other methods. In Fig. 5, we generated ROC curves for association strength of 0.3 and 0.5. Fig. 5a,b show the results with a priori hierarchical clustering and Fig. 5c,d is with no such preprocessing steps. Using hierarchical clustering we could correctly find three clusters of gene traits at cutoff $0.8$. In Fig. 5, when association strength is small (i.e., 0.3), AML and A+$\ell_1/\ell_2$ significantly outperformed other methods. As association strength increased, the performance of multi-task learning methods improved quickly while methods based on a single trait such as Lasso and Single SNP showed gradual increase of performance.

We computed test errors on 100 simulated dataset using 30 samples for test and 84 samples for training. On average, AML achieved the best test error rate of 0.9427, and the order of other methods in terms of test errors is: A + $\ell_1/\ell_2$ (0.9506), SML (1.0436), $\ell_1/\ell_\infty$ (1.0578) and Lasso (1.1080).

## 5 Yeast eQTL dataset

We analyze the yeast eQTL dataset [16] that contains expression levels of 5,637 genes and 2,956 SNPs. The genotype data include genetic variants of 114 yeast strains that are progenies of the standard laboratory strain (BY) and a wild strain (RM). We used 141 modules given by Lee et al. [3] as groups of gene traits, and extracted unique 1,260 SNPs from 2,956 SNPs for our analysis. For prior biological knowledge on SNPs used for adaptive multi-task Lasso, we downloaded 12 features from Saccharomyces Genome Database (http://www.yeastgenome.org) including 11 discrete and 1 continuous feature (conservation score). For a discrete feature, we set its value as $f_t^j = s(2)$ if the feature is found on the $j$-th SNP, $f_t^j = s(1)$ otherwise. For conservation score, we set $f_t^j = s(\text{score})$. All the features are then standardized.

Fig. 6 represents $\omega$ learned from the yeast eQTL dataset ($\nu$ is almost identical to $\omega$). The features are ncRNA ($f_1$), noncoding exon ($f_2$), snRNA ($f_3$), tRNA ($f_4$), intron ($f_5$), binding site ($f_6$), 5' UTR intron ($f_7$), LTR retrotransposon ($f_8$), ARS ($f_9$), snoRNA ($f_{10}$), transposable element gene ($f_{11}$) and conservation score ($f_{12}$). Five discrete features turn out to be important including ncRNA, snRNA, binding site, 5' UTR intron and snoRNA as well as one continuous feature, i.e., conservation score. These results agree with biological insights. For example, ncRNA, snRNA and snoRNA are potentially important for gene regulation since they are functional RNA molecules having

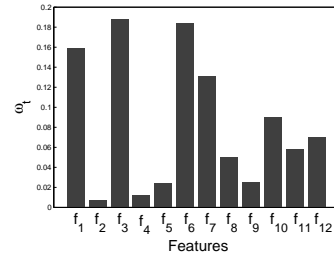

Figure 6: Learned weights of $\omega$ on the yeast eQTL dataset.

a variety of roles such as transcriptional regulation [18]. Also, conservation score would be significant since mutation in conserved region is more likely to result in phenotypic effects.

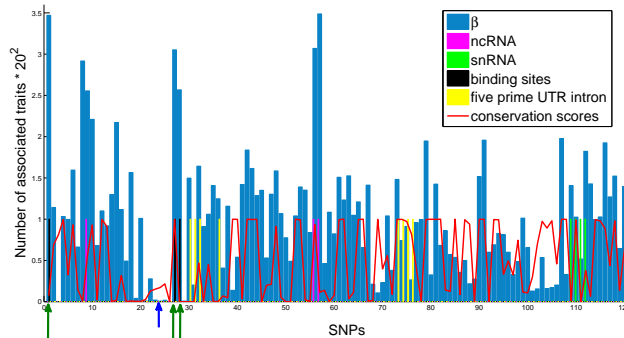

Figure 7: Plot of 121 SNPs on chromosome 1 and 2 vs the number of genes affected by the SNPs from the yeast eQTL analysis (blue bar). Five significant prior knowledge on SNPs are overlapped with the plot. For the four discrete priors (ncRNA, snRNA, binding site, 5' UTR intron) we set the value to 1 if annotated, 0 otherwise. Binding sites and regions with no associated traits are denoted by long green and short blue arrows.

Fig. 7 shows the number of associated genes for SNPs on chromosome 1 and 2, superimposed on 5 significant features. We see that association mapping results were affected by both priors and data. For example, genomic region indicated by blue arrow shows weak association with traits, where conservation score is low and no other annotations exist. Also we can see that three SNPs located on binding sites affect a larger number of gene traits (see green arrows). As an example of biological analysis, we investigate these three association SNPs. The three SNPs are located on telomeres (chr1:483, chr1:229090, chr2:9425 (chromosome:coordinate)), and these genomic locations are in cis to Abf1p (autonomously replicating sequence binding factor-1) binding sites. In biology, it is known that Abf1p acts as a global transcriptional regulator in yeast [19]. Thus, the genomic regions in telomeres would be good candidates for novel putative eQTL hotspots that regulate the expression levels of many genes. They were not reported as eQTL hotspots in Yvert et al. [20].

## 6 Conclusions

In this paper, we proposed a novel regularized regression model, referred to as adaptive multi-task Lasso, which takes into account multiple traits simultaneously while weights of different covariates are learned adaptively from prior knowledge and data. Our simulation results support that our model outperforms other methods via $\ell_1$ and $\ell_1/\ell_2$ penalty over multiple related genes, and especially adaptively learned regularization significantly improved the performance. In our experiments on the yeast eQTL dataset, we could identify putative three eQTL hotspots with biological supports where SNPs are associated with a large number of genes.

**Acknowledgments**

This work was done under a support from NIH 1 R01 GM087694-01, NIH 1RC2HL101487-01 (ARRA), AFOSR FA9550010247, ONR N0001140910758, NSF Career DBI-0546594, NSF IIS-0713379 and Alfred P. Sloan Fellowship awarded to E.X.

# References

[1] R. Sladek, G. Rocheleau, J. Rung, C. Dina, L. Shen, D. Serre, P. Boutin, D. Vincent, A. Belisle, S. Hadjadj, et al. A genome-wide association study identifies novel risk loci for type 2 diabetes. *Nature*, 445(7130):881–885, 2007.

[2] R. Tibshirani. Regression shrinkage and selection via the Lasso. *Journal of the Royal Statistical Society. Series B (Methodological)*, 58(1):267–288, 1996.

[3] S.I. Lee, A.M. Dudley, D. Drubin, P.A. Silver, N.J. Krogan, D. Pe'er, and D. Koller. Learning a prior on regulatory potential from eQTL data. *PLoS Genetics*, 5(1):e1000358, 2009.

[4] S. Kim and E. P. Xing. Statistical estimation of correlated genome associations to a quantitative trait network. *PLoS Genetics*, 5(8):e1000587, 2009.

[5] G. Obozinski, B. Taskar, and M. Jordan. Multi-task feature selection. In *Technical Report, Department of Statistics, University of California, Berkeley*, 2006.

[6] M. Szafranski, Y. Grandvalet, and P. Morizet-Mahoudeaux. Hierarchical penalization. *Advances in Neural Information Processing Systems*, 20:1457–1464, 2007.

[7] H. Zou. The adaptive Lasso and its oracle properties. *Journal of the American Statistical Association*, 101(476):1418–1429, 2006.

[8] S.I. Lee, V. Chatalbashev, D. Vickrey, and D. Koller. Learning a meta-level prior for feature relevance from multiple related tasks. In *Proceedings of the 24th International Conference on Machine Learning*, pages 489–496, 2007.

[9] T. Park and G. Casella. The bayesian Lasso. *Journal of the American Statistical Association*, 103(482):681–686, 2008.

[10] B. M. Marlin, M. Schmidt, and K. P. Murphy. Group sparse priors for covariance estimation. In *Proceedings of the 25th Conference on Uncertainty in Artificial Intelligence*, pages 383–392, 2009.

[11] E. Gómez, M. A. Gomez-Viilegas, and J. M. Marin. A multivariate generalization of the power exponential family of distributions. *Communications in Statistics-Theory and Methods*, 27(3):589–600, 1998.

[12] H. Lee, A. Battle, R. Raina, and A. Y. Ng. Efficient sparse coding algorithms. *Advances in Neural Information Processing Systems*, 19:801–808, 2007.

[13] J. Duchi, S. Shalev-Shwartz, Y. Singer, and T. Chandra. Efficient projections onto the $\ell_1$-ball for learning in high dimensions. In *Proceedings of the 25th International Conference on Machine Learning*, pages 272–279, 2008.

[14] J. Friedman, T. Hastie, and R. Tibshirani. A note on the group Lasso and a sparse group Lasso. *arXiv:1001.0736v1 [math.ST]*, 2010.

[15] T. T. Wu and K. Lange. Coordinate descent algorithms for Lasso penalized regression. *Ann. Appl. Stat*, 2(1):224–244, 2008.

[16] R. B. Brem and L. Kruglyak. The landscape of genetic complexity across 5,700 gene expression traits in yeast. *Proceedings of the National Academy of Sciences of the United States of America*, 102(5):1572–1577, 2005.

[17] S. Purcell, B. Neale, K. Todd-Brown, L. Thomas, M. A. R. Ferreira, D. Bender, J. Maller, P. Sklar, P. I. W. De Bakker, M. J. Daly, et al. PLINK: a tool set for whole-genome association and population-based linkage analyses. *The American Journal of Human Genetics*, 81(3):559–575, 2007.

[18] G. Storz. An expanding universe of noncoding RNAs. *Science*, 296(5571):1260–1263, 2002.

[19] T. Miyake, J. Reese, C. M. Loch, D. T. Auble, and R. Li. Genome-wide analysis of ARS (autonomously replicating sequence) binding factor 1 (Abf1p)-mediated transcriptional regulation in Saccharomyces cerevisiae. *Journal of Biological Chemistry*, 279(33):34865–34872, 2004.

[20] G. Yvert, R. B. Brem, J. Whittle, J. M. Akey, E. Foss, E. N. Smith, R. Mackelprang, L. Kruglyak, et al. Trans-acting regulatory variation in Saccharomyces cerevisiae and the role of transcription factors. *Nature Genetics*, 35(1):57–64, 2003.

